# Gene-Gene Relationship Modeling Based on Genetic Evidence for Single-Cell RNA-Seq Data Imputation

**Daeho Um**\*
Samsung Advanced Institute of Technology (SAIT)
daeho.um@samsung.com

**Ji Won Yoon**
Chung-Ang University
jiwonyoon@cau.ac.kr

**Seong Jin Ahn**
Korea Advanced Institute of Science and Technology (KAIST)
sja1015@kaist.ac.kr

**Yunha Yeo**
Korea University
serinahyeo@korea.ac.kr

## Abstract

Single-cell RNA sequencing (scRNA-seq) technologies enable the exploration of cellular heterogeneity and facilitate the construction of cell atlases. However, scRNA-seq data often contain a large portion of missing values (false zeros) or noisy values, hindering downstream analyses. To recover these false zeros, propagation-based imputation methods have been proposed using $k$-NN graphs. However they model only associating relationships among genes within a cell, while, according to well-known genetic evidence, there are both associating and dissociating relationships among genes. To apply this genetic evidence to gene-gene relationship modeling, this paper proposes a novel imputation method that newly employs dissociating relationships in addition to associating relationships. Our method constructs a $k$-NN graph to additionally model dissociating relationships via the negation of a given cell-gene matrix. Moreover, our method standardizes the value distribution (mean and variance) of each gene to have standard distributions regardless of the gene. Through extensive experiments, we demonstrate that the proposed method achieves exceptional performance gains over state-of-the-art methods in both cell clustering and gene expression recovery across six scRNA-seq datasets, validating the significance of using complete gene-gene relationships in accordance with genetic evidence. The source code is available at https://github.com/daehoum1/scCR.

## 1 Introduction

Single-cell RNA sequencing (scRNA-seq) has become one of the most widely used technologies in biomedical research due to its ability to measure genome-wide gene expression at the single-cell level [1–3]. ScRNA-seq enables us to discover novel cell types [4], analyze cellular trajectories [5], and improve understanding human disease [6, 7]. However, scRNA-seq analysis encounters significant challenges due to the high rate of zero values in scRNA-seq data represented by a cell-gene matrix. Specifically, owing to the low RNA capture rate, scRNA-seq data often contain zero values. These zero values represent unobserved gene expression resulting from both technical omissions (referred to as *dropouts* [8]) and true biological absence. Moreover, even non-zero values in scRNA-seq data suffer from various sources of noise, such as cell cycle effects and batch effects [9, 10].

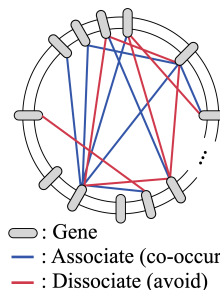

: Gene
— : Associate (co-occur)
— : Dissociate (avoid)

Figure 1: Within a cell, there are two types of relationships among genes.

---

To deal with the missing or noisy gene expression in scRNA-seq data, diverse imputation methods have been proposed, which can be categorized into non-graph-based, graph neural network (GNN)-based, and propagation-based methods. Among these methods, propagation-based methods [11, 12] have been favored due to their outstanding performance. The propagation-based methods construct a $k$-nearest neighbor ($k$-NN) graph on scRNA-seq data represented as a cell-gene matrix, and fill in missing values by propagating nonzero values on the $k$-NN graph. Despite their effectiveness, they overlook well-known genetic evidence [13, 14], which means that there are two types of relationships between genes: associating relationship and dissociating one. As shown in Figure 1, associating relationships represent genes that co-occur, whereas dissociating relationships represent genes that avoid co-occurrence.

However, the existing methods cannot model the dissociating gene-gene relationship by constructing a simple $k$-NN graph to connect only the associating genes with similar occurrence patterns. Consequently, these methods fail to connect dissociating genes. Within a cell, when considering the value to be imputed for gene Q, the value for its associating gene can assist in inferring the value for gene Q. However, its dissociating gene can also provide crucial information: if its dissociating gene has a high value, the value for gene Q may be low, as they tend to avoid each other. Additionally, the value distribution of a gene often differs significantly from that of other genes [15]. Therefore, the sum of propagated values from other genes may lead to the mixing of values at various scales, which is not suitable for data recovery.

To resolve the aforementioned problems, we propose a novel propagation-based imputation scheme called Single-Cell Complete Relationship (scCR) for scRNA-seq data, which models both associating and dissociating gene-gene relationships. scCR concatenates a given cell-gene matrix and its negation, then standardizes the value distribution (mean and variance) in each column (*i.e.*, gene) of the concatenated matrix. Subsequently, we construct a $k$-NN graph on this concatenated and standardized matrix to connect both associating and dissociating genes within a cell. Through a propagation process on this $k$-NN graph, scCR effectively denoises scRNA-seq data by capturing complete gene-gene relationships. Extensive experimental results demonstrate that scCR significantly outperforms state-of-the-art methods in both gene expression recovery and cell clustering. Through experiment, we further confirm that scCR can model dissociating gene-gene relationships inherent in scRNA-seq data.

The main contributions of our work are summarized as follows: (1) We newly propose an effective imputation method for scRNA-seq data, which is based on the genetic evidence. Our method can model complete gene-gene relationships, including both associating and dissociating relationships; (2) We employ a standardization step before propagation among genes for additional performance improvement in downstream tasks on scRNA-seq data; (3) By modeling dissociating gene-gene relationships and utilizing the standardization step, our scCR significantly improves performance in various downstream tasks, outperforming the state-of-the-art methods by a large margin.

## 2   Related Work

**Handling noise in scRNA-seq data.** Approaches for handling noise in scRNA-seq data can be categorized into non-graph-based, GNN-based, and propagation-based methods. As pioneering efforts to impute zero values, non-graph-based methods predominantly employ either statistical techniques [16, 17] or autoencoder frameworks [17, 18]. Building on this foundation, graph-based approaches, including GNN-based and propagation-based methods, have received significant attention due to their ability to model relationships among cells and genes through graph structures. scGNN [19] leverages cell-cell relationships by constructing a cell-cell similarity matrix within a graph autoencoder framework. scGCL [20] is a graph autoencoder framework that exploits contrastive learning to capture cell-cell relationships. scTAG [21] is a clustering method that employs a graph autoencoder framework using a cell-cell $k$-NN graph, which jointly optimizes clustering loss and reconstruction loss.

**Propagation-based imputation in scRNA-seq data.** Propagation-based imputation methods have shown their superiority in scRNA-seq data imputation. They promote greater similarity in gene expression among cells that are already similar through iterative propagation steps. While MAGIC [22] utilizes a diffusion mechanism to denoise scRNA-seq data, updating values through the diffusion of both zero values and observed nonzero values may be significantly affected by false zero values (*i.e.*, dropouts). To address this issue, Feature Propagation (FP) [23] can be a good solution because FP preserves observed values during diffusion while updates unobserved values through

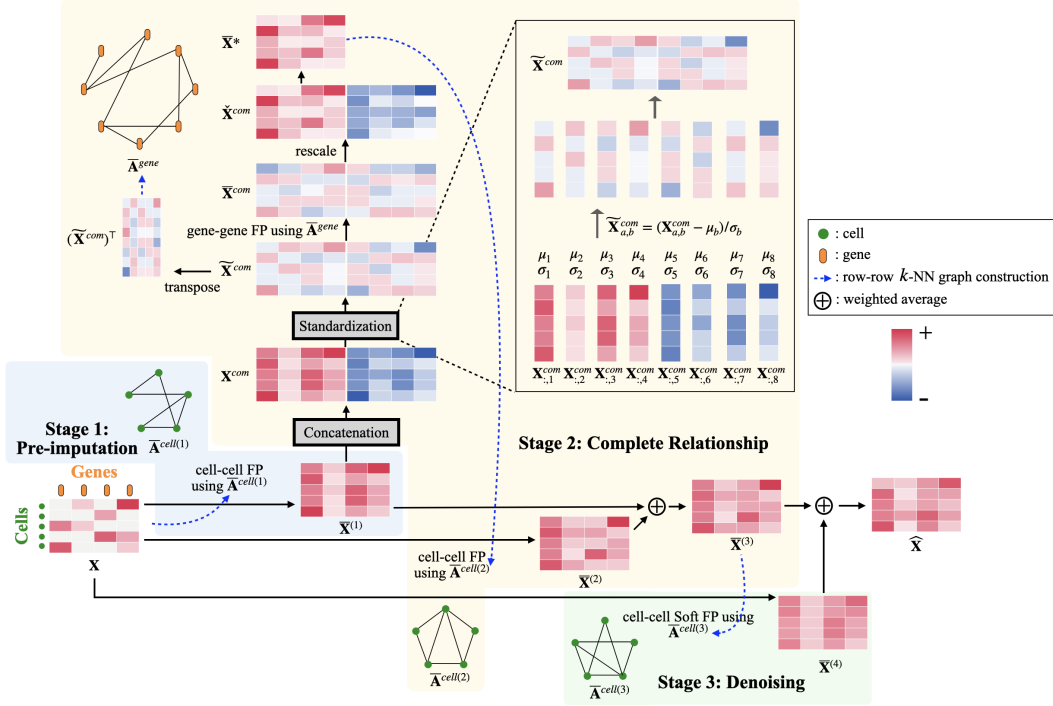

Figure 2: A brief overview of Single-Cell Complete Relationship (scCR).

diffusion. Propagation-based imputation methods have shown their superiority in scRNA-seq data imputation. scFP [11] adopts FP developed for graph-structured data to resolve imputation for scRNA-seq data. scFP constructs a cell-cell $k$-NN graph and applies FP for the imputation of zero values. Very recently, [12] proposes scBFP to utilize gene-gene relationships as well as cell-cell relationships. scBFP consists of two stages, and in each stage, it applies FP using a gene-gene $k$-NN graph and a cell-cell $k$-NN graph, respectively. Although scBFP is designed to leverage gene-gene relationships, the simple addition of FP using a gene-gene $k$-NN graph cannot effectively exploit gene-gene relationships due to the following two reasons: **(1)** a gene-gene $k$-NN graph can connect only associating genes which have co-occurrence relationships while overlooking the presence of dissociating gene-gene relationships; **(2)** Since the distributions for each gene significantly varies, propagation without additional processing will degrade recovery performance.

## 3   Preliminaries

**Notation.** A graph can be represented as $\mathcal{G} = (\mathcal{V}, \mathcal{E})$, where $\mathcal{V} = \{v_1, \ldots, v_N\}$ is the set of $N$ nodes and $\mathcal{E}$ is the set of edges. The connectivity of $\mathcal{G}$ can be represented by the adjacency matrix $\mathbf{A} \in \{0, 1\}^{N \times N}$ with $\mathbf{A}_{i,j} = 1$ iff $(v_i, v_j) \in \mathcal{E}$ and $\mathbf{A}_{i,j} = 0$ otherwise. Given an arbitrary matrix $\mathbf{B} \in \mathbb{R}^{a \times b}$, we let $k\text{NN}(\cdot) : \mathbb{R}^{a \times b} \to \{\mathbb{R}\}^{a \times a}$ be a function that generates a normalized adjacency matrix $\overline{\mathbf{A}}$ of the row-row $k$-NN graph based on cosine similarity. Here, the normalized adjacency matrix $\overline{\mathbf{A}}$ is obtained by $\overline{\mathbf{A}} = \mathbf{D}^{-1/2} \mathbf{A} \mathbf{D}^{-1/2}$ where $\mathbf{A}$ is the adjacency matrix of the $k$-NN graph and $\mathbf{D}$ is a degree matrix with diagonal entries $\mathbf{D}_{i,i} = \sum_j \mathbf{A}_{i,j}$. Consequently, while $k\text{NN}(\mathbf{X})$ yields $\overline{\mathbf{A}}^{cell} \in \{0, 1\}^{C \times C}$ of the cell-cell $k$-NN graph from $\mathbf{X}$, $k\text{NN}(\mathbf{X}^\top)$ produces $\overline{\mathbf{A}}^{gene} \in \{0, 1\}^{G \times G}$ of the gene-gene $k$-NN graph from $\mathbf{X}$. We let $\mathbf{B}_{i,:}$ and $\mathbf{B}_{:,j}$ denote the $i$-th row vector of $\mathbf{B}$ and the $j$-th column vector of $\mathbf{B}$, respectively.

**Feature Propagation.** FP-based algorithms [23, 24] are proposed to impute missing features in graph-structured data. The core idea of FP is to impute missing values by diffusing observed values while preserving these observed values. Assume that a given graph $\mathcal{G} = (\mathcal{V}, \mathcal{E})$ has a feature matrix $\mathbf{X} \in \mathbb{R}^{N \times F}$ with missing values, where rows and columns correspond to nodes and $F$ feature channels, respectively. We use $\overline{\mathbf{A}} \in \mathbb{R}^{N \times N}$ to denote a normalized adjacency matrix of the graph. To preserve known features during the diffusion process, we mark the positions of the features to be

preserved with 1 in the mask matrix $\mathbf{M} \in \{0, 1\}^{N \times F}$; here, values of 1 in $\mathbf{M}$ indicate the location of observed features. We express FP as a function by $\overline{\mathbf{X}} = \text{FP}(\mathbf{X}, \overline{\mathbf{A}}, \mathbf{M})$ where $\overline{\mathbf{X}} \in \mathbb{R}^{N \times F}$ is an output matrix. A detailed explanation of FP is provided in Appendix A. In summary, FP fills in missing values in $\mathbf{X}$ through diffusion using $\mathbf{A}$ while preserving features corresponding to values of 1 in $\mathbf{M}$. It is noteworthy that $\text{FP}(\mathbf{X}, \mathbf{A}, \mathbf{M})$ performs propagation among the rows of $\mathbf{X}$. In scRNA-seq data imputation, FP-based imputation methods treat zero values as missing values to be imputed via features diffused from non-zero values.

# 4 Proposed Method

## 4.1 Overview of scCR

In this paper, we design a novel imputation framework for scRNA-seq data, namely scCR, which utilizes complete gene-gene relationships. Unlike existing work, scCR exploits both associating and dissociating relationships, which contain valuable biological information. Given highly noisy scRNA-seq data, especially having a high number of false zeros, the goal of scCR is to recover scRNA-seq data by imputing zero values. As shown in Figure 2, our proposed framework consists of three stages: pre-imputation, complete relation, and denoising. Throughout these three stages, we enhance a gene expression matrix by gradually integrating complete gene-gene and cell-cell relationships.

## 4.2 Pre-Imputation Stage

We consider a cell-gene matrix $\mathbf{X} \in \mathbb{R}^{C \times G}$, where $C$ and $G$ represent the number of cells and genes, respectively. We let $\mathbf{A}^{cell} \in \{0, 1\}^{C \times C}$ denote an adjacency matrix of a cell-cell graph. Similarly, we let $\mathbf{A}^{gene} \in \{0, 1\}^{G \times G}$ denote an adjacency matrix of a gene-gene graph. Building a $k$-NN graph directly on $\mathbf{X}$ can lead to performance degradation in downstream tasks due to the noisy nature of $\mathbf{X}$. Therefore, scCR begins with the pre-imputation stage, which creates a pre-imputed matrix to be used for $k$-NN graph construction in the complete relationship stage. In this stage, scCR imputes zero values through intercellular (*i.e.*, cell-cell) propagation.

**Cell-cell FP.** We first construct a cell-cell $k$-NN graph by $\overline{\mathbf{A}}^{cell(1)} = k\text{NN}(\mathbf{X})$. scCR then employs FP to impute zero values by the diffusion of nonzero values among cells. We let $\mathbf{M}^{\mathbf{X}} \in \{0, 1\}^{C \times G}$ be a mask matrix with $\mathbf{M}^{\mathbf{X}}_{i,j} = 1$ iff $\mathbf{X}_{i,j} \neq 1$ and $\mathbf{M}^{\mathbf{X}}_{i,j} = 0$ otherwise, which indicates the positions of the nonzero features in $\mathbf{X}$ to be preserved during the diffusion. Cell-cell FP using $\overline{\mathbf{A}}^{cell(1)}$ is performed as follows,

$$\overline{\mathbf{X}}^{(1)} = \text{FP}(\mathbf{X}, \overline{\mathbf{A}}^{cell(1)}, \mathbf{M}^{\mathbf{X}}) \tag{1}$$

where $\overline{\mathbf{X}}^{(1)} \in \mathbb{R}^{C \times G}$ is an output of the pre-imputation stage, which is utilized in the following complete relationship stage.

## 4.3 Complete Relationship Stage

In the complete relationship stage, we refine $\overline{\mathbf{X}}^{(1)}$ through gene-gene and cell-cell propagation. ScBFP [12] adopts gene-gene FP on the $k$-NN graph constructed based on cosine similarity. However, it overlooks two key issues: **(1)** The similarity-based gene-gene $k$-NN graph can connect only associating (or co-occurring) genes, excluding highly correlated dissociating (or avoiding) genes, which can offer important biological information for imputation. This occurs because associating genes may have high cosine similarity due to their co-occurrence. These genes with high cosine similarity may become connected in the cosine-similarity-based $k$-NN graph . **(2)** As shown in Figure 3, since each gene has the distinct distribution within a gene expression matrix [15], the value distribution for a gene varies significantly among genes. Although imputation methods for scRNA-seq generally normalize a gene expression matrix in a cell-wise manner, varying scales across genes still remain after the cell-wise normalization. Therefore, propagation across genes may degrade accurate recovery by mixing values with different scales.

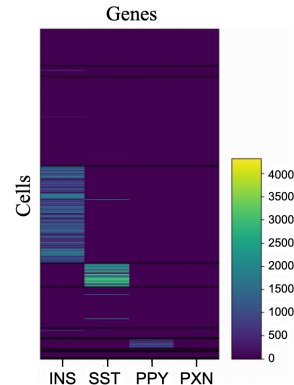

Figure 3: A subset of the gene expression matrix in the Baron Human dataset.

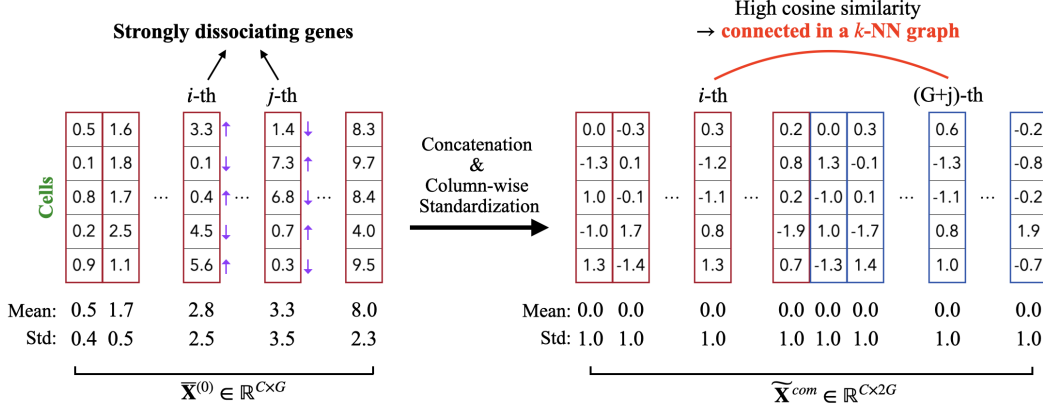

Figure 4: An illustration of the concatenation and standardization processes in the complete relationship stage. Std denotes standard deviation.

**Concatenation.** To address these key issues, we propose a novel propagation scheme called gene-gene standardized FP. The gene-gene standardized FP first produces $\mathbf{X}^{com} \in \mathbb{R}^{C \times 2G}$ by concatenating $\overline{\mathbf{X}}^{(1)}$ and its negative matrix along columns, *i.e.*, $\mathbf{X}^{com} = [\overline{\mathbf{X}}^{(1)}, -\overline{\mathbf{X}}^{(1)}]$.

**Standardization.** Subsequently, to enable every gene to have the same scale during propagation among genes, we standardize $\mathbf{X}^{com}$ in a column-wise manner. For $b \in \{1, \ldots, 2G\}$, we standardize $\mathbf{X}^{com}$ to $\widetilde{\mathbf{X}}^{com} \in \mathbb{R}^{C \times 2G}$ as follows,

$$\widetilde{\mathbf{X}}^{com}_{a,b} = \frac{(\mathbf{X}^{com}_{a,b} - \mu_b)}{\sigma_b} \text{ where } \mu_b = \sum_{a=1}^{C} \mathbf{X}^{com}_{a,b}, \ \sigma_b = \sqrt{\frac{1}{C-1} \sum_{a=1}^{C} (\mathbf{X}^{com}_{a,b} - \mu_b)^2}. \quad (2)$$

Here, $\mu_b$ and $\sigma_b$ denote the mean and standard deviation of the $b$-th column (*i.e.*, gene) of $\mathbf{X}^{com}$. Since all the columns in $\widetilde{\mathbf{X}}^{com}$ are standardized, SFP can effectively perform propagation-based imputation without the mixing of values at various scales, addressing the aforementioned issue **(2)**. Furthermore, by concatenating $\overline{\mathbf{X}}^{(1)}$ with its negative matrix before the construction of $\widetilde{\mathbf{X}}^{com}$, SFP can connect not only associating but also dissociating gene-gene relationships. As demonstrated in Figure 4, assume that the $i$-th gene and the $j$-th gene have strong dissociating relationships, where $i, j \in \{1, \ldots, G\}$. Within any $a$-th cell ($a \in \{1, \ldots, C\}$), $\overline{\mathbf{X}}^{(1)}_{a,i}$ will be very high when $\overline{\mathbf{X}}^{(1)}_{a,j}$ is very low and vice versa. After the standardization, the cosine similarity between the $i$-th gene and the $j$-th gene will have a large negative value, which cannot be connected in a $k$-NN graph. However, through the concatenation, $\widetilde{\mathbf{X}}^{com}_{:,(G+j)}$ corresponds to $-\overline{\mathbf{X}}^{(1)}_{:,j}$. Thus, the cosine similarity between $\widetilde{\mathbf{X}}^{com}_{:,i}$ and $\widetilde{\mathbf{X}}^{com}_{:,(G+j)}$ has a large positive value, which will be connected in a $k$-NN graph. Therefore, through gene-gene propagation using this $k$-NN graph constructed by using $\widetilde{\mathbf{X}}^{com}$, we can effectively exploit complete gene-gene relationships, including associating and dissociating relationships.

**Gene-gene FP.** We build a gene-gene $k$-NN graph on $(\widetilde{\mathbf{X}}^{com})^{\top}$ by $\overline{\mathbf{A}}^{gene} = k\text{NN}((\widetilde{\mathbf{X}}^{com})^{\top})$ where $\overline{\mathbf{A}}^{gene} \in \mathbb{R}^{2G \times 2G}$. We then perform the gene-gene standardized FP using $\overline{\mathbf{A}}^{gene}$ as follows:

$$\overline{\mathbf{X}}^{com} = (\text{FP}((\widetilde{\mathbf{X}}^{com})^{\top}, \mathbf{A}^{gene}, [\mathbf{M}^{\mathbf{X}}, \mathbf{M}^{\mathbf{X}}]^{\top}))^{\top} \quad (3)$$

where $\overline{\mathbf{X}}^{com} \in \mathbb{R}^{C \times 2G}$. Since each gene in $\overline{\mathbf{X}}^{com}$ does not have its original scale due to the standardization, we return all the columns in $\overline{\mathbf{X}}^{com}$ to their original scale as follows:

$$\check{\mathbf{X}}^{com}_{a,b} = \sigma_b \overline{\mathbf{X}}^{com}_{a,b} + \mu_b \quad (4)$$

where $\check{\mathbf{X}}^{com} \in \mathbb{R}^{C \times 2G}$ is the rescaled matrix. We then reduce $\check{\mathbf{X}}^{com}$ by retaining the first $G$ columns, and we denote this reduced matrix as $\overline{\mathbf{X}}^{*} \in \mathbb{R}^{C \times G}$. $\overline{\mathbf{X}}^{*}$ is a final output of the gene-gene standardized FP.

**Cell-cell FP.** $\overline{\mathbf{X}}^{*}$ containing information from complete gene-gene relationships plays a crucial role in scCR by contributing the formation of all subsequent cell-gene matrices. To perform additional

Table 1: Performance on cell clustering, measured by ARI, NMI, and CA. Standard deviation errors are given. Figures highlighted in green indicate performance improvements over the most competitive baseline at each setting.

| Dataset | Baron Mouse | | | Pancreas | | | Mouse Bladder | | |
|---|---|---|---|---|---|---|---|---|---|
| Method | ARI | NMI | CA | ARI | NMI | CA | ARI | NMI | CA |
| scTAG | $0.565_{\pm0.016}$ | $0.689_{\pm0.023}$ | $0.526_{\pm0.163}$ | $0.678_{\pm0.141}$ | $0.789_{\pm0.011}$ | $0.69_{\pm0.108}$ | $0.604_{\pm0.149}$ | $0.734_{\pm0.047}$ | $0.605_{\pm0.011}$ |
| DCA | $0.447_{\pm0.022}$ | $0.710_{\pm0.010}$ | $0.562_{\pm0.002}$ | $0.566_{\pm0.002}$ | $0.786_{\pm0.001}$ | $0.727_{\pm0.002}$ | $0.447_{\pm0.022}$ | $0.710_{\pm0.010}$ | $0.562_{\pm0.002}$ |
| AutoClass | $0.408_{\pm0.002}$ | $0.699_{\pm0.002}$ | $0.525_{\pm0.004}$ | $0.564_{\pm0.020}$ | $0.795_{\pm0.009}$ | $0.724_{\pm0.030}$ | $0.506_{\pm0.02}$ | $0.732_{\pm0.009}$ | $0.613_{\pm0.029}$ |
| scGNN 2.0 | $0.441_{\pm0.021}$ | $0.734_{\pm0.029}$ | $0.575_{\pm0.019}$ | $0.562_{\pm0.054}$ | $0.793_{\pm0.049}$ | $0.728_{\pm0.061}$ | $0.488_{\pm0.041}$ | $0.717_{\pm0.015}$ | $0.595_{\pm0.033}$ |
| scGCL | $0.478_{\pm0.001}$ | $0.720_{\pm0.000}$ | $0.645_{\pm0.003}$ | $0.645_{\pm0.061}$ | $0.755_{\pm0.042}$ | $0.747_{\pm0.026}$ | $0.529_{\pm0.002}$ | $0.725_{\pm0.005}$ | $0.598_{\pm0.008}$ |
| MAGIC | $0.419_{\pm0.007}$ | $0.712_{\pm0.007}$ | $0.557_{\pm0.015}$ | $0.595_{\pm0.007}$ | $0.803_{\pm0.004}$ | $0.765_{\pm0.022}$ | $0.565_{\pm0.004}$ | $0.754_{\pm0.001}$ | $0.651_{\pm0.005}$ |
| scFP | $0.613_{\pm0.000}$ | $0.817_{\pm0.000}$ | $0.763_{\pm0.000}$ | $0.802_{\pm0.001}$ | $0.872_{\pm0.000}$ | $0.878_{\pm0.000}$ | $0.655_{\pm0.002}$ | $0.767_{\pm0.000}$ | $0.730_{\pm0.001}$ |
| scBFP | $0.660_{\pm0.000}$ | $0.813_{\pm0.001}$ | $0.763_{\pm0.001}$ | $\mathbf{0.864}_{\pm0.000}$ | $\mathbf{0.900}_{\pm0.001}$ | $\mathbf{0.918}_{\pm0.006}$ | $0.694_{\pm0.000}$ | $0.761_{\pm0.002}$ | $\mathbf{0.779}_{\pm0.001}$ |
| scCR (Ours) | $\mathbf{0.827}_{\pm0.139}$ | $\mathbf{0.847}_{\pm0.034}$ | $\mathbf{0.846}_{\pm0.084}$ | $0.812_{\pm0.000}$ | $0.855_{\pm0.000}$ | $0.873_{\pm0.000}$ | $\mathbf{0.704}_{\pm0.000}$ | $\mathbf{0.778}_{\pm0.000}$ | $0.765_{\pm0.000}$ |
| | (+25.3%) | (+3.7%) | (+10.9%) | - | - | - | (+1.7%) | (+1.4%) | - |

| Dataset | Zeisel | | | Worm Neuron | | | Baron Human | | |
|---|---|---|---|---|---|---|---|---|---|
| Method | ARI | NMI | CA | ARI | NMI | CA | ARI | NMI | CA |
| scTAG | $0.723_{\pm0.010}$ | $0.716_{\pm0.013}$ | $0.712_{\pm0.029}$ | $0.532_{\pm0.134}$ | $0.641_{\pm0.007}$ | $0.439_{\pm0.003}$ | $0.612_{\pm0.029}$ | $0.718_{\pm0.028}$ | $0.610_{\pm0.158}$ |
| DCA | $0.693_{\pm0.005}$ | $0.739_{\pm0.005}$ | $0.764_{\pm0.004}$ | $0.502_{\pm0.017}$ | $0.690_{\pm0.016}$ | $0.700_{\pm0.031}$ | $0.545_{\pm0.001}$ | $0.763_{\pm0.004}$ | $0.558_{\pm0.001}$ |
| AutoClass | $0.673_{\pm0.006}$ | $0.714_{\pm0.009}$ | $0.746_{\pm0.006}$ | $0.488_{\pm0.002}$ | $0.668_{\pm0.001}$ | $0.699_{\pm0.001}$ | $0.523_{\pm0.02}$ | $0.743_{\pm0.005}$ | $0.553_{\pm0.023}$ |
| scGNN 2.0 | $0.533_{\pm0.050}$ | $0.657_{\pm0.063}$ | $0.666_{\pm0.041}$ | $0.453_{\pm0.061}$ | $0.637_{\pm0.03}$ | $0.653_{\pm0.051}$ | $0.525_{\pm0.031}$ | $0.744_{\pm0.025}$ | $0.569_{\pm0.014}$ |
| scGCL | $0.663_{\pm0.003}$ | $0.715_{\pm0.116}$ | $0.717_{\pm0.001}$ | $0.601_{\pm0.014}$ | $0.676_{\pm0.005}$ | $0.754_{\pm0.012}$ | $0.593_{\pm0.027}$ | $0.744_{\pm0.056}$ | $0.671_{\pm0.077}$ |
| MAGIC | $0.696_{\pm0.003}$ | $0.747_{\pm0.002}$ | $0.765_{\pm0.002}$ | $0.512_{\pm0.014}$ | $0.719_{\pm0.006}$ | $0.770_{\pm0.013}$ | $0.562_{\pm0.012}$ | $0.788_{\pm0.007}$ | $0.596_{\pm0.012}$ |
| scFP | $0.848_{\pm0.000}$ | $0.812_{\pm0.000}$ | $0.886_{\pm0.000}$ | $0.524_{\pm0.330}$ | $\mathbf{0.731}_{\pm0.014}$ | $0.766_{\pm0.031}$ | $0.676_{\pm0.000}$ | $0.826_{\pm0.000}$ | $0.732_{\pm0.000}$ |
| scBFP | $0.835_{\pm0.000}$ | $0.792_{\pm0.000}$ | $0.869_{\pm0.000}$ | $\mathbf{0.608}_{\pm0.000}$ | $0.715_{\pm0.000}$ | $\mathbf{0.792}_{\pm0.000}$ | $0.677_{\pm0.000}$ | $0.827_{\pm0.000}$ | $0.733_{\pm0.000}$ |
| scCR (Ours) | $\mathbf{0.902}_{\pm0.000}$ | $\mathbf{0.863}_{\pm0.000}$ | $\mathbf{0.952}_{\pm0.000}$ | $0.520_{\pm0.014}$ | $0.711_{\pm0.006}$ | $0.746_{\pm0.012}$ | $\mathbf{0.823}_{\pm0.000}$ | $\mathbf{0.858}_{\pm0.000}$ | $\mathbf{0.827}_{\pm0.000}$ |
| | (+6.4%) | (+6.3%) | (+7.5%) | - | - | - | (+21.6%) | (+3.8%) | (+12.8%) |

cell-cell FP using complete gene-gene relationships inherent in $\overline{\mathbf{X}}^*$, we construct cell-cell a $k$-NN graph by $\overline{\mathbf{A}}^{cell(2)} = k\mathrm{NN}(\mathbf{X}^*)$. We perform cell-cell FP using $\mathbf{X}$ and $\overline{\mathbf{A}}^{cell(2)}$ as follows,

$$\overline{\mathbf{X}}^{(2)} = \mathrm{FP}(\mathbf{X}, \overline{\mathbf{A}}^{cell(2)}, \mathbf{M}^{\mathbf{X}}) \tag{5}$$

where $\overline{\mathbf{X}}^{(2)}$ is an output of this cell-cell FP.

**Weighted sum.** $\overline{\mathbf{X}}^{(3)}$, which is a final output of complete relationship stage is produced by the weighted sum of $\overline{\mathbf{X}}^{(2)}$ and $\overline{\mathbf{X}}^{(1)}$ as follows:

$$\overline{\mathbf{X}}^{(3)} = \alpha\overline{\mathbf{X}}^{(1)} + (1 - \alpha)\overline{\mathbf{X}}^{(2)} \tag{6}$$

where $0 < \alpha < 1$ is a hyperparameter. $\overline{\mathbf{X}}^{(3)}$ can incorporate valuable biological information since complete gene-gene relationships are delivered by $\overline{\mathbf{X}}^{(2)}$.

### 4.4 Denoising Stage

**Cell-cell Soft FP.** While the pre-imputation and complete relationship stages focus on imputing zero values, the denoising stage aims to remove noise in the overall values of $\mathbf{X}$ via propagation-based smoothing. To exploit complete gene-gene relationships, the denoising stage utilizes $\overline{\mathbf{X}}^{(3)}$ containing them. We first build a cell-cell $k$-NN graph by $\overline{\mathbf{A}}^{cell(3)} = k\mathrm{NN}(\overline{\mathbf{X}}^{(3)})$. To denoise $\mathbf{X}$, we adopt Soft FP [11] that does not maintain zero values during propagation. We apply Soft FP [11] to $\mathbf{X}$ as follows,

$$\overline{\mathbf{X}}^{(4)}(t) = \beta\overline{\mathbf{A}}^{cell(3)}\overline{\mathbf{X}}^{(4)}(t - 1) + (1 - \beta)\mathbf{X}, \ \ t = 1, \cdots, K, \tag{7}$$

where $K$ is the total number of propagation steps, $\overline{\mathbf{X}}^{(4)}(0) = \mathbf{X}$, $\overline{\mathbf{X}}^{(4)}(t) \in \mathbb{R}^{C \times G}$ is the updated cell-gene matrix after $t$ propagation steps, and $0 < \beta < 1$ is a hyperparameter. An output of the denoising stage, denoted by $\overline{\mathbf{X}}^{(4)}(K)$, is obtained after $K$ steps.

**Weighted sum.** The final output of our scCR, $\widehat{\mathbf{X}}$, is the weighted sum of $\overline{\mathbf{X}}^{(3)}$ and $\overline{\mathbf{X}}^{(4)}(K)$ as follows:

$$\widehat{\mathbf{X}} = \gamma\overline{\mathbf{X}}^{(3)} + (1 - \gamma)\overline{\mathbf{X}}^{(4)}(K). \tag{8}$$

where and $0 < \gamma \le 1$ is a hyperparameter. In summary, unlike existing propagation-based methods, our scCR enables the use of complete gene-gene relationships in denoising scRNA-seq.

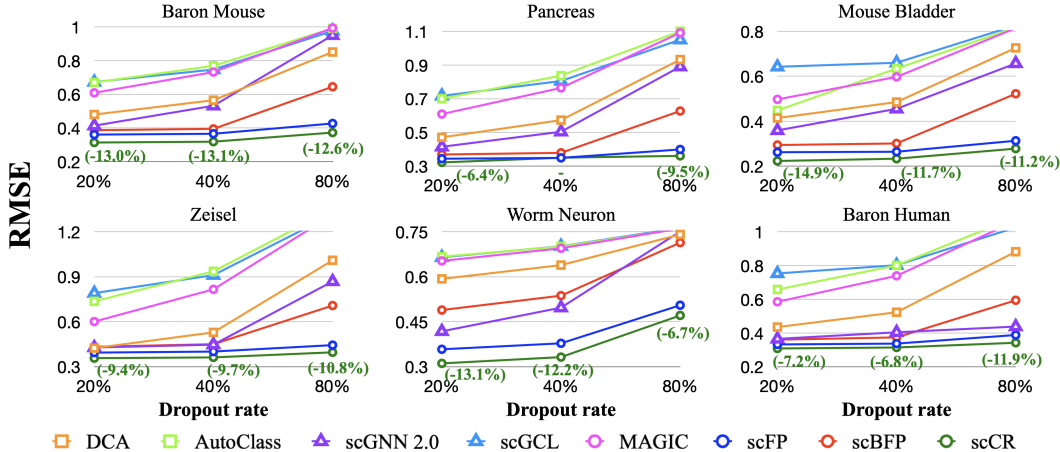

Figure 5: Performance on dropout recovery, measured by RMSE. Figures highlighted in green indicate reduction rates from the most competitive baseline at each setting.

# 5 Experiments

## 5.1 Experimental Setup

We performed comparative evaluation of scCR on six widely used scRNA-seq datasets with gold-standard cell type information: Baron Mouse [25], Pancreas [26], Mouse Bladder [27], Zeisel [2], Worm Neuron [28], and Baron Human [25]. We compared our scCR with eight state-of-the-art methods handling noise in scRNA-seq data: (1) non-graph-based methods: DCA [17] and AutoClass [18]; (2) GNN-based methods: scTAG [21], scGNN 2.0 [19], and scGCL [20]; (3) propagation-based methods: MAGIC [22], scFP [11], and scBFP [12]. We evaluated scTAG only on cell clustering, since scTAG is a clustering method. To evaluate the cell clustering of scCR and baselines, we utilized three standard evaluation metrics: Adjusted Rand Index (ARI), Normalized Mutual Information (NMI), and Clustering Accuracy (CA). We used KMeans clustering to compare the clustering performance of various imputation methods, including our scCR. For dropout recovery, we employed two standard evaluation metrics: Root Mean Square Error (RMSE) and Median L1 Distance.

**Implementation.** For fair comparisons, we set the hyperparameters of baselines according to specifications in the papers and official codes. We reported the average performance across three independent runs. Experimental details regarding datasets, baselines, evaluation metrics, and hyperparameter settings are provided in Appendix G.

## 5.2 Results

**scCR enables improved cell clustering.** To validate the effectiveness of scCR in cell clustering, we evaluated its clustering performance. Table 1 presents the performance comparison. While FP-based baselines, including scFP and scBFP, outperformed other baselines, scCR delivered the best or competitive cell clustering performance across all datasets. Specifically, scCR improved ARI by $25.3\%$, $1.7\%$, $6.4\%$, and $21.6\%$ compared to previous state-of-the-art results on Baron Mouse, Mouse Bladder, Zeisel, and Baron Human, respectively.

**scCR effectively recovers dropout values.** Since dropouts can occur at various rates, we generated false zeros (*i.e.*, dropouts) at non-zero values in datasets by applying varying dropout rates. As shown in Figure 5, scCR significantly improved dropout recovery performance in various dropout rates across the datasets. We confirmed that scCR effectively reduced RMSE between imputed values and their original values with large reduction rates, only except for the Pancreas dataset with $40\%$ dropout. We provided a recovery performance comparison, measured by the median L1 distance in Appendix H.2, which also demonstrates the effectiveness of scCR.

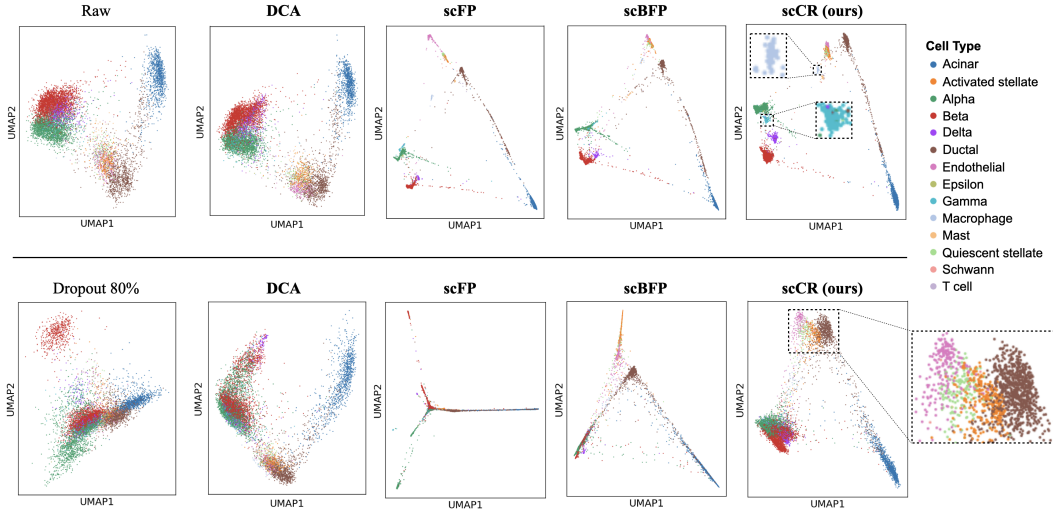

Figure 6: UMAP visualization using the Baron Human dataset, comparing scCR with the three most competitive imputation methods. The first row shows the visualization of the raw data and their imputed results. The second row displays the visualization of data subjected to an $80\%$ dropout rate and their imputed results.

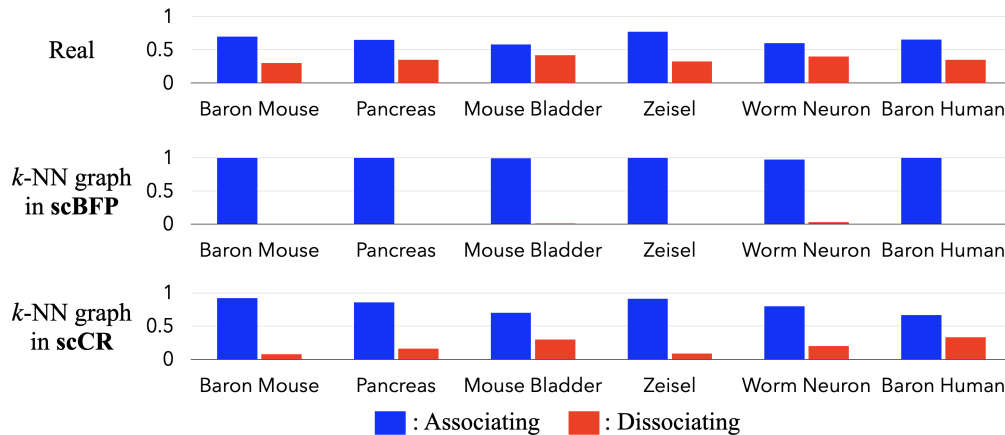

Figure 7: The first row indicates the percentages of associating and dissociating gene-gene relationships in datasets. The second and third rows represents the percentages of associating and dissociating relationships within a gene-gene $k$-NN graph in each method.

**scCR identifies rare cell types well.** To verify the effectiveness of scCR in identifying rare cell types, we conducted in-depth analysis using the Baron Human dataset. We visualized the two-dimensional UMAP [29] repesentations of the raw data and the data with an $80\%$ dropout rate applied. As shown in the first row in Figure 6, scCR effectively identified rare cell types with few cells, such as 'gamma' and 'Macrophage'. It is noteworthy that even under severe dropouts, scCR separated cell types well (in the second row), whereas compared methods failed.

**Does scCR really model dissociating relationships?** To show that scCR models both associating and dissociating gene-gene relationships, we first investigated the ratio of dissociating relationships to associating relationships. For this investigation, we define that associating or dissociating relationships exist when the absolute values of the correlation coefficients between genes exceed the top $25\%$. We determine that a positive sign in the correlation coefficients indicates associating relationships, while a negative sign indicates dissociating relationships. We then measured the percentages of

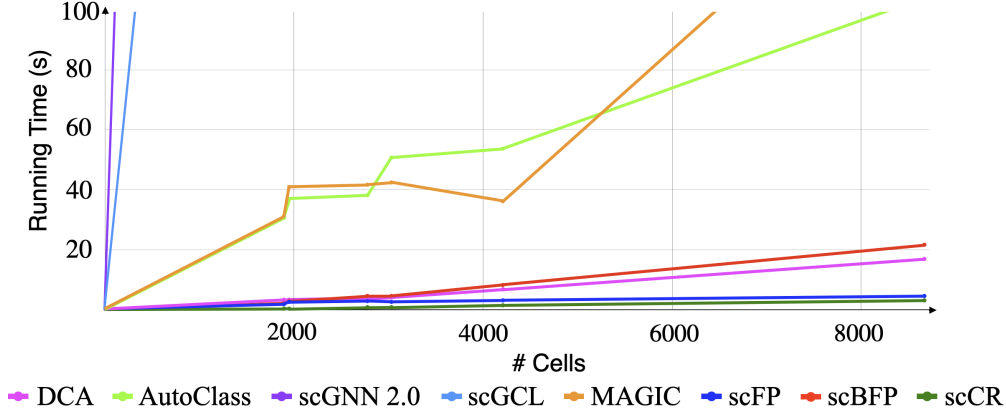

Figure 8: Running time comparison of scCR and baselines according to the number of cells.

associating and dissociating relationships within gene-gene $k$-NN graphs in scBFP and our scCR. As shown in Figure 7, scBFP hardly models dissociating gene-gene relationships. In contrast, scCR effectively models dissociating relationships, enabling the use of complete gene-gene relationships in its imputation.

**scCR is even faster than existing imputation methods.** To show the advantage of scCR in terms of imputation time, we measured the running time of scCR and imputation methods on datasets. As shown in Figure 8, scCR showed the lowest running time across all the datasets, regardless of the number of cells.

An ablation study (See Appendix C), further experimental results (See Appendix H), and the proof of convergence of FP (See Appendix B) are provided in Appendix.

# 6    Conclusion

In this paper, we proposed a novel imputation framework called Single-Cell Complete Relationship (scCR) for scRNA-seq data imputation. scCR utilized complete gene-gene relationships by concatenating a given cell-gene matrix with its negation and facilitated effective gene-gene propagation through the standardization of the cell-gene matrix in a gene-wise manner. These processes, grounded in genetic evidence, led to significant performance improvements over state-of-the-art methods in various downstream tasks on scRNA-seq data, with fast imputation times. Furthermore, our work is not limited to simply utilizing genetic evidence to design a framework. We validated this evidence within real scRNA-seq datasets, and confirmed that our scCR effectively leveraged this insight. Like other scRNA-seq imputation methods, scCR is specifically designed for scRNA-seq data. However, since scRNA-seq data are inherently matrix-formatted, scCR can be extended to general tabular data imputation. We expect that the concatenation process will be effective even for general tabular data, as there are often both positive and negative correlation coefficients between channels in such data. The extension of scCR to other domains is left for future work.

# Broader Impacts

Our work provides an important insight that, when applying machine learning to the biomedical domain, it is crucial to approach with biological grounding rather than focusing solely on applying existing cutting-edge machine learning techniques. Since scRNA-seq has opened a new frontier for understanding biological systems [15, 30, 31], we believe that our work will contribute to the biomedical domain by enhancing the analysis of human diseases and the discovery of new genetic observations. We have not identified any negative impacts of our work on society.

## Acknowledgements

This work was supported by Institute of Information & communications Technology Planning & Evaluation (IITP) grant funded by the Korea government(MSIT) (2021-0-01341,Artificial Intelligence Graduate School Program(Chung-Ang University)).

## Footnotes

[2]https://github.com/Junseok0207/scFP

## References

[1] Evan Z Macosko, Anindita Basu, Rahul Satija, James Nemesh, Karthik Shekhar, Melissa Goldman, Itay Tirosh, Allison R Bialas, Nolan Kamitaki, Emily M Martersteck, et al. Highly parallel genome-wide expression profiling of individual cells using nanoliter droplets. *Cell*, 161 (5):1202–1214, 2015.

[2] Amit Zeisel, Ana B Muñoz-Manchado, Simone Codeluppi, Peter Lönnerberg, Gioele La Manno, Anna Juréus, Sueli Marques, Hermany Munguba, Liqun He, Christer Betsholtz, et al. Cell types in the mouse cortex and hippocampus revealed by single-cell rna-seq. *Science*, 347(6226): 1138–1142, 2015.

[3] Michael JT Stubbington, Orit Rozenblatt-Rosen, Aviv Regev, and Sarah A Teichmann. Single-cell transcriptomics to explore the immune system in health and disease. *Science*, 358(6359): 58–63, 2017.

[4] Hadas Keren-Shaul, Amit Spinrad, Assaf Weiner, Orit Matcovitch-Natan, Raz Dvir-Szternfeld, Tyler K Ulland, Eyal David, Kuti Baruch, David Lara-Astaiso, Beata Toth, et al. A unique microglia type associated with restricting development of alzheimer's disease. *Cell*, 169(7): 1276–1290, 2017.

[5] Cole Trapnell, Davide Cacchiarelli, Jonna Grimsby, Prapti Pokharel, Shuqiang Li, Michael Morse, Niall J Lennon, Kenneth J Livak, Tarjei S Mikkelsen, and John L Rinn. Pseudo-temporal ordering of individual cells reveals dynamics and regulators of cell fate decisions. *Nature biotechnology*, 32(4):381, 2014.

[6] Monika M Gladka, Bas Molenaar, Hesther De Ruiter, Stefan Van Der Elst, Hoyee Tsui, Danielle Versteeg, Grègory PA Lacraz, Manon MH Huibers, Alexander Van Oudenaarden, and Eva Van Rooij. Single-cell sequencing of the healthy and diseased heart reveals cytoskeleton-associated protein 4 as a new modulator of fibroblasts activation. *Circulation*, 138(2):166–180, 2018.

[7] William Stephenson, Laura T Donlin, Andrew Butler, Cristina Rozo, Bernadette Bracken, Ali Rashidfarrokhi, Susan M Goodman, Lionel B Ivashkiv, Vivian P Bykerk, Dana E Orange, et al. Single-cell rna-seq of rheumatoid arthritis synovial tissue using low-cost microfluidic instrumentation. *Nature communications*, 9(1):791, 2018.

[8] Stephanie C Hicks, F William Townes, Mingxiang Teng, and Rafael A Irizarry. Missing data and technical variability in single-cell rna-sequencing experiments. *Biostatistics*, 19(4):562–578, 2018.

[9] Florian Buettner, Kedar N Natarajan, F Paolo Casale, Valentina Proserpio, Antonio Scialdone, Fabian J Theis, Sarah A Teichmann, John C Marioni, and Oliver Stegle. Computational analysis of cell-to-cell heterogeneity in single-cell rna-sequencing data reveals hidden subpopulations of cells. *Nature biotechnology*, 33(2):155–160, 2015.

[10] Uri Shaham, Kelly P Stanton, Jun Zhao, Huamin Li, Khadir Raddassi, Ruth Montgomery, and Yuval Kluger. Removal of batch effects using distribution-matching residual networks. *Bioinformatics*, 33(16):2539–2546, 2017.

[11] Sukwon Yun, Junseok Lee, and Chanyoung Park. Single-cell rna-seq data imputation using feature propagation. *arXiv preprint arXiv:2307.10037*, 2023.

[12] Junseok Lee, Sukwon Yun, Yeongmin Kim, Tianlong Chen, Manolis Kellis, and Chanyoung Park. Single-cell rna sequencing data imputation using bi-level feature propagation. *Briefings in Bioinformatics*, 25(3):bbae209, 2024.

[13] Fiona Jane Whelan, Martin Rusilowicz, and James Oscar McInerney. Coinfinder: detecting significant associations and dissociations in pangenomes. *Microbial genomics*, 6(3):e000338, 2020.

[14] Rebecca J Hall, Fiona J Whelan, Elizabeth A Cummins, Christopher Connor, Alan McNally, and James O McInerney. Gene-gene relationships in an escherichia coli accessory genome are linked to function and mobility. *Microbial Genomics*, 7(9):000650, 2021.

[15] Anoop P Patel, Itay Tirosh, John J Trombetta, Alex K Shalek, Shawn M Gillespie, Hiroaki Wakimoto, Daniel P Cahill, Brian V Nahed, William T Curry, Robert L Martuza, et al. Single-cell rna-seq highlights intratumoral heterogeneity in primary glioblastoma. *Science*, 344(6190): 1396–1401, 2014.

[16] Wei Vivian Li and Jingyi Jessica Li. An accurate and robust imputation method scimpute for single-cell rna-seq data. *Nature communications*, 9(1):997, 2018.

[17] Gökcen Eraslan, Lukas M Simon, Maria Mircea, Nikola S Mueller, and Fabian J Theis. Single-cell rna-seq denoising using a deep count autoencoder. *Nature communications*, 10(1):390, 2019.

[18] Hui Li, Cory R Brouwer, and Weijun Luo. A universal deep neural network for in-depth cleaning of single-cell rna-seq data. *Nature Communications*, 13(1):1901, 2022.

[19] Juexin Wang, Anjun Ma, Yuzhou Chang, Jianting Gong, Yuexu Jiang, Ren Qi, Cankun Wang, Hongjun Fu, Qin Ma, and Dong Xu. scgnn is a novel graph neural network framework for single-cell rna-seq analyses. *Nature communications*, 12(1):1882, 2021.

[20] Zehao Xiong, Jiawei Luo, Wanwan Shi, Ying Liu, Zhongyuan Xu, and Bo Wang. scgcl: an imputation method for scrna-seq data based on graph contrastive learning. *Bioinformatics*, 39 (3):btad098, 2023.

[21] Zhuohan Yu, Yifu Lu, Yunhe Wang, Fan Tang, Ka-Chun Wong, and Xiangtao Li. Zinb-based graph embedding autoencoder for single-cell rna-seq interpretations. In *Proceedings of the AAAI conference on artificial intelligence*, volume 36, pages 4671–4679, 2022.

[22] David Van Dijk, Roshan Sharma, Juozas Nainys, Kristina Yim, Pooja Kathail, Ambrose J Carr, Cassandra Burdziak, Kevin R Moon, Christine L Chaffer, Diwakar Pattabiraman, et al. Recovering gene interactions from single-cell data using data diffusion. *Cell*, 174(3):716–729, 2018.

[23] Emanuele Rossi, Henry Kenlay, Maria I Gorinova, Benjamin Paul Chamberlain, Xiaowen Dong, and Michael M Bronstein. On the unreasonable effectiveness of feature propagation in learning on graphs with missing node features. In *Learning on Graphs Conference*, pages 11–1. PMLR, 2022.

[24] Daeho Um, Jiwoong Park, Seulki Park, and Jin young Choi. Confidence-based feature imputation for graphs with partially known features. In *The Eleventh International Conference on Learning Representations*, 2023. URL https://openreview.net/forum?id=YPKBIILy-Kt.

[25] Maayan Baron, Adrian Veres, Samuel L Wolock, Aubrey L Faust, Renaud Gaujoux, Amedeo Vetere, Jennifer Hyoje Ryu, Bridget K Wagner, Shai S Shen-Orr, Allon M Klein, et al. A single-cell transcriptomic map of the human and mouse pancreas reveals inter-and intra-cell population structure. *Cell systems*, 3(4):346–360, 2016.

[26] Malte D Luecken, Maren Büttner, Kridsadakorn Chaichoompu, Anna Danese, Marta Interlandi, Michaela F Müller, Daniel C Strobl, Luke Zappia, Martin Dugas, Maria Colomé-Tatché, et al. Benchmarking atlas-level data integration in single-cell genomics. *Nature methods*, 19(1): 41–50, 2022.

[27] Xiaoping Han, Renying Wang, Yincong Zhou, Lijiang Fei, Huiyu Sun, Shujing Lai, Assieh Saadatpour, Ziming Zhou, Haide Chen, Fang Ye, et al. Mapping the mouse cell atlas by microwell-seq. *Cell*, 172(5):1091–1107, 2018.

[28] Junyue Cao, Jonathan S Packer, Vijay Ramani, Darren A Cusanovich, Chau Huynh, Riza Daza, Xiaojie Qiu, Choli Lee, Scott N Furlan, Frank J Steemers, et al. Comprehensive single-cell transcriptional profiling of a multicellular organism. *Science*, 357(6352):661–667, 2017.

[29] Leland McInnes, John Healy, Nathaniel Saul, and Lukas Großberger. Umap: Uniform manifold approximation and projection. *Journal of Open Source Software*, 3(29), 2018.

[30] Diego Adhemar Jaitin, Ephraim Kenigsberg, Hadas Keren-Shaul, Naama Elefant, Franziska Paul, Irina Zaretsky, Alexander Mildner, Nadav Cohen, Steffen Jung, Amos Tanay, et al. Massively parallel single-cell rna-seq for marker-free decomposition of tissues into cell types. *Science*, 343(6172):776–779, 2014.

[31] Nicholas Schaum, Jim Karkanias, Norma F Neff, Andrew P May, Stephen R Quake, Tony Wyss-Coray, Spyros Darmanis, Joshua Batson, Olga Botvinnik, Michelle B Chen, et al. Single-cell transcriptomics of 20 mouse organs creates a tabula muris: The tabula muris consortium. *Nature*, 562(7727):367, 2018.

[32] Abraham Berman and Robert J Plemmons. *Nonnegative matrices in the mathematical sciences*. SIAM, 1994.

[33] Fan RK Chung. *Spectral graph theory*, volume 92. American Mathematical Soc., 1997.

[34] Yunqing Liu, Jiayi Zhao, Taylor S Adams, Ningya Wang, Jonas C Schupp, Weimiao Wu, John E McDonough, Geoffrey L Chupp, Naftali Kaminski, Zuoheng Wang, et al. idesc: identifying differential expression in single-cell rna sequencing data with multiple subjects. *BMC bioinformatics*, 24(1):318, 2023.

[35] F Alexander Wolf, Philipp Angerer, and Fabian J Theis. Scanpy: large-scale single-cell gene expression data analysis. *Genome biology*, 19:1–5, 2018.

## A  Feature Propagation

Assume that a given graph $\mathcal{G} = (\mathcal{V}, \mathcal{E})$ has a feature matrix $\mathbf{X} \in \mathbb{R}^{N \times F}$ with missing values, where rows and columns correspond to nodes and $F$ feature channels, respectively. We use $\overline{\mathbf{A}} \in \mathbb{R}^{N \times N}$ to denote a normalized adjacency, which is an input for feature propagation (FP). To preserve known features during the diffusion process, we mark the positions of the features to be preserved with $1$ in the mask matrix $\mathbf{M} \in \{0, 1\}^{N \times F}$. Here, values of $1$ in $\mathbf{M}$ indicate the location of observed features, where feature values will be preserved.

To formally explain the diffusion process of FP in detail, we temporarily reorder nodes for notational convenience. Since observed values may differ across channels, we reorder the nodes for each channel. When we consider the $a$-th channel, based on the values of $1$ in $\mathbf{M}_{:,a}$, we let $\mathcal{V}_k^a$ be the set of nodes whose $a$-th values are known (observed). Similarly, by examining zero values in $\mathbf{M}_{:,a}$, $\mathcal{V}_u^a$ denotes the set of nodes whose $a$-th values are unknown (missing). By reordering the nodes in the order of $\mathcal{V}_k^a$ and $\mathcal{V}_u^a$, the entire feature values and the adjacency matrix for the $a$-th channel can be expressed as

$$\mathbf{x}^a = \begin{bmatrix} \mathbf{x}_k^a \\ \mathbf{x}_u^a \end{bmatrix}, \qquad \overline{\mathbf{A}}^{(a)} = \begin{bmatrix} \overline{\mathbf{A}}_{kk}^{(a)} & \overline{\mathbf{A}}_{ku}^{(a)} \\ \overline{\mathbf{A}}_{uk}^{(a)} & \overline{\mathbf{A}}_{uu}^{(a)} \end{bmatrix}, \tag{9}$$

where $\mathbf{x}^a$ is a column vector representing features for the $a$-th channel in $\mathbf{X}$ in the order of $\mathcal{V}_k^a$ and $\mathcal{V}_u^a$. Here, $\mathbf{x}_k^a \in \mathbb{R}^{|\mathcal{V}_k^a|}$ and $\mathbf{x}_u^a \in \mathbb{R}^{|\mathcal{V}_u^a|}$. Similarly, $\overline{\mathbf{A}}^{(a)}$ consists of four sub-matrices related to $\mathcal{V}_k^a$ and $\mathcal{V}_u^a$. It is noteworthy that although $\overline{\mathbf{A}}^{(a)} \in \mathbb{R}^{N \times N}$ and $\overline{\mathbf{A}} \in \mathbb{R}^{N \times N}$ is different due to reordering, they represent the same graph structure.

To preserve observed values during the diffusion process, we replace the first $|\mathcal{V}_k^a|$ rows in $\overline{\mathbf{A}}^{(a)}$ with one-hot vectors indicating $\mathcal{V}_k^a$. Consequently, we obtain a transition matrix $\widetilde{\mathbf{A}}^{(a)} \in \mathbb{R}^{N \times N}$ expressed by

$$\widetilde{\mathbf{A}}^{(a)} = \begin{bmatrix} \mathbf{I}_{kk} & \mathbf{0}_{ku} \\ \mathbf{A}_{uk}^{(a)} & \mathbf{A}_{uu}^{(a)} \end{bmatrix}, \tag{10}$$

where $\mathbf{I}_{nn} \in \mathbb{R}^{|\mathcal{V}_n^a| \times |\mathcal{V}_n^a|}$ is an identity matrix and $\mathbf{0}_{nz} \in \mathbb{R}^{|\mathcal{V}_n^a| \times |\mathcal{V}_z^a|}$ is a zero matrix. The diffusion process of FP is implemented by iterative propagation steps utilizing $\widetilde{\mathbf{A}}^{(a)}$ as

$$\begin{aligned} \bar{\mathbf{x}}^a(t) &= \widetilde{\mathbf{A}}^{(a)} \bar{\mathbf{x}}^a(t-1), \ \ t = 1, \cdots, K; \\ \bar{\mathbf{x}}^a(0) &= \begin{bmatrix} \mathbf{x}_k^a \\ \mathbf{0}_u^a \end{bmatrix}, \end{aligned} \tag{11}$$

where $\bar{\mathbf{x}}^a(t)$ is an imputed feature vector after $t$ propagation steps and $\mathbf{0}_u^a$ denotes a zero vector of the same length as $|\mathcal{V}_k^a|$. As $K \to \infty$, this recursion converges and $\bar{\mathbf{x}}^a(t)$ reaches a steady state (the proof can be found in Appendix B). We use $\bar{\mathbf{x}}^a(K)$ with large enough $K$ to approximate the steady state.

After this diffusion process for the entire channels, we attain $\{\bar{\mathbf{x}}^a(K)\}_{a=1}^F$. Since these vectors have different ordering due to channel-wise reordering, we rearrange $\{\bar{\mathbf{x}}^a(K)\}_{a=1}^F$ in the original order and construct $\overline{\mathbf{X}} \in \mathbb{R}^{N \times F}$ by stacking the originally ordered vectors in $\{\bar{\mathbf{x}}^a(K)\}_{a=1}^F$ along the channels. In summary, FP fills in missing values in $\mathbf{X}$ through diffusion using $\overline{\mathbf{A}}$ while preserving features corresponding to values of $1$ in $\mathbf{M}$.

## B  Proof of Convergence of Feature Propagation

Feature propagation (FP) [23] utilize symmetrically normalized transition matrix for the diffusion process implemented by iterative propagation steps. We prove the convergence of this diffusion process as follows.

**Proposition 1.** *The transition matrix for the $a$-th channel is defined by*

$$\widetilde{\mathbf{A}}^{(a)} = \begin{bmatrix} \mathbf{I}_{kk} & \mathbf{0}_{ku} \\ \mathbf{A}_{uk}^{(a)} & \mathbf{A}_{uu}^{(a)} \end{bmatrix},$$

where $\widetilde{\mathbf{A}}^{(a)}$ is symmetrically normalized. Using $\widetilde{\mathbf{A}}^{(a)}$, the diffusion process in the $a$-th channel is defined by

$$\bar{\mathbf{x}}^a(t) = \widetilde{\mathbf{A}}^{(a)}\bar{\mathbf{x}}^a(t-1), \ \ t = 1, \cdots, K;$$

$$\bar{\mathbf{x}}^a(0) = \begin{bmatrix} \mathbf{x}_k^a \\ \mathbf{0}_u^a \end{bmatrix},$$

Then, $\lim_{K \to \infty} \bar{\mathbf{x}}^{(a)}(K)$ converges.

The proof of Proposition 1 refers to [23]. We begin with two lemmas.

**Lemma 1.** $\overline{\mathbf{A}}^{(a)}$ is the symmetrically normalized matrix calculated by $\overline{\mathbf{A}}^{(a)} = (\mathbf{D}^{(a)})^{-1/2}\mathbf{A}^{(a)}(\mathbf{D}^{(a)})^{-1/2}$ where $\mathbf{D}^{(a)}$ is a diagonal matrix that has diagonal entities $\mathbf{D}_{ii}^{(a)} = \sum_j \mathbf{A}_{i,j}^{(a)}$. $\overline{\mathbf{A}}_{uu}^{(a)}$ is the $|\bar{\mathbf{x}}_u^{(a)}| \times |\bar{\mathbf{x}}_u^{(a)}|$ bottom-right submatrix of $\overline{\mathbf{A}}^{(a)}$ and let $\rho(\cdot)$ denote spectral radius. Then, $\rho(\overline{\mathbf{A}}_{uu}^{(a)}) < 1$.

*Proof.* Consider $\overline{\mathbf{A}}_{uu0}^{(a)} \in \mathbb{R}^{N \times N}$, where the bottom right submatrix is equal to $\overline{\mathbf{A}}_{uu}^{(a)}$ and all other elements are zero. That is,

$$\overline{\mathbf{A}}_{uu0}^{(a)} = \begin{bmatrix} \mathbf{0}_{kk} & \mathbf{0}_{ku} \\ \mathbf{0}_{uk} & \overline{\mathbf{A}}_{uu}^{(a)} \end{bmatrix}$$

where $\mathbf{0}_{kk} \in \{0\}^{|\bar{\mathbf{x}}_k^{(a)}| \times |\bar{\mathbf{x}}_k^{(a)}|}$, $\mathbf{0}_{ku} \in \{0\}^{|\bar{\mathbf{x}}_k^{(a)}| \times |\bar{\mathbf{x}}_u^{(a)}|}$, and $\mathbf{0}_{uk} \in \{0\}^{|\bar{\mathbf{x}}_u^{(a)}| \times |\bar{\mathbf{x}}_k^{(a)}|}$. Given that $\overline{\mathbf{A}}^{(a)}$ represents the weighted adjacency matrix of the connected graph $\mathcal{G}$, $\overline{\mathbf{A}}_{uu0}^{(a)} \leq \overline{\mathbf{A}}^{(a)}$ element-wise and $\overline{\mathbf{A}}_{uu0}^{(a)} \neq \overline{\mathbf{A}}^{(a)}$. Furthermore, considering that $\overline{\mathbf{A}}_{uu0}^{(a)} + \overline{\mathbf{A}}^{(a)}$ constitutes the weighted adjacency matrix of a strongly connected graph, we can conclude that $\overline{\mathbf{A}}_{uu0}^{(a)} + \overline{\mathbf{A}}^{(a)}$ is irreducible based on Theorem 2.2.7 in [32]. Consequently, applying Corollary 2.1.5 in [32], $\rho(\overline{\mathbf{A}}_{uu0}^{(a)}) < \rho(\overline{\mathbf{A}}^{(a)})$. Furthermore, $\rho(\overline{\mathbf{A}}^{(a)}) \leq 1$ since we can write $\overline{\mathbf{A}}^{(a)} = \mathbf{I} - (\mathbf{D}^{(a)})^{-1/2}\mathbf{A}(\mathbf{D}^{(a)})^{-1/2}$, where $(\mathbf{D}^{(a)})^{-1/2}\mathbf{A}(\mathbf{D}^{(a)})^{-1/2}$ has eigenvalues in the range $[0, 2]$ [33]. Note that since both $\overline{\mathbf{A}}_{uu0}^{(a)}$ and $\overline{\mathbf{A}}_{uu}^{(a)}$ share the same non-zero eigenvalues, it follows that $\rho(\overline{\mathbf{A}}_{uu0}^{(a)}) = \rho(\overline{\mathbf{A}}_{uu}^{(a)})$. Ultimately, combining these inequalities leads to the result $\rho(\overline{\mathbf{A}}_{uu}^{(a)}) = \rho(\overline{\mathbf{A}}_{uu0}^{(a)}) < \rho(\overline{\mathbf{A}}^{(a)}) = 1$. $\square$

**Lemma 2.** $\mathbf{I}_{uu} - \overline{\mathbf{A}}_{uu}^{(a)}$ is invertible where $\mathbf{I}_{uu}$ is the $|\bar{\mathbf{x}}_u^{(a)}| \times |\bar{\mathbf{x}}_u^{(a)}|$ identity matrix.

*Proof.* Since 1 is not an eigenvalue of $\overline{\mathbf{A}}_{uu}^{(a)}$ by Lemma 1, 0 is not an eigenvlaue of $\mathbf{I}_{uu} - \overline{\mathbf{A}}_{uu}^{(a)}$. Thus $\mathbf{I}_{uu} - \overline{\mathbf{A}}_{uu}^{(a)}$ is invertible. $\square$

We now prove Propostion 1 as follows.

*Proof.* The recursive relation can be written as

$$\bar{\mathbf{x}}^{(a)}(t) = \begin{bmatrix} \bar{\mathbf{x}}_k^{(a)}(t) \\ \bar{\mathbf{x}}_u^{(a)}(t) \end{bmatrix} = \begin{bmatrix} \mathbf{I}_{kk} & \mathbf{0}_{ku} \\ \overline{\mathbf{A}}_{uk}^{(a)} & \overline{\mathbf{A}}_{uu}^{(a)} \end{bmatrix} \begin{bmatrix} \bar{\mathbf{x}}_k^{(a)}(t-1) \\ \bar{\mathbf{x}}_u^{(a)}(t-1) \end{bmatrix} = \begin{bmatrix} \bar{\mathbf{x}}_k^{(a)}(t-1) \\ \overline{\mathbf{A}}_{uk}^{(a)}\bar{\mathbf{x}}_k^{(a)}(t-1) + \overline{\mathbf{A}}_{uu}^{(a)}\bar{\mathbf{x}}_u^{(a)}(t-1) \end{bmatrix}.$$

Since $\bar{\mathbf{x}}_k^{(a)}(t) = \bar{\mathbf{x}}_k^{(a)}(t-1)$ in the first $|\bar{\mathbf{x}}_k^{(a)}|$ rows, it follows that $\bar{\mathbf{x}}_k^{(a)}(K) = \ldots = \bar{\mathbf{x}}_k^{(a)}$. That is, $\bar{\mathbf{x}}_k^{(a)}(K)$ retains the values of $\mathbf{x}_k^{(a)}$. Therefore, $\lim_{K \to \infty} \bar{\mathbf{x}}_k^{(a)}(K)$ converges to $\mathbf{x}_k^{(a)}$.

Table 2: Ablation study. Con and Sta denote the concatenation and standardization, respectively, which enable capturing dissociating gene-gene relationships.

| Con | Sta | Baron Mouse | | | Zeisel | | | Baron Human | | |
|---|---|---|---|---|---|---|---|---|---|---|
| | | ARI | NMI | CA | ARI | NMI | CA | ARI | NMI | CA |
| ✗ | ✗ | $0.625_{\pm0.001}$ | $0.777_{\pm0.001}$ | $0.719_{\pm0.003}$ | $0.789_{\pm0.000}$ | $0.738_{\pm0.000}$ | $0.851_{\pm0.000}$ | $0.805_{\pm0.000}$ | $0.837_{\pm0.000}$ | $0.813_{\pm0.000}$ |
| ✗ | ✓ | $0.623_{\pm0.000}$ | $0.788_{\pm0.001}$ | $0.710_{\pm0.001}$ | $0.898_{\pm0.000}$ | $0.856_{\pm0.000}$ | $0.948_{\pm0.000}$ | $0.816_{\pm0.000}$ | $0.846_{\pm0.002}$ | $0.819_{\pm0.001}$ |
| ✓ | ✗ | $0.629_{\pm0.001}$ | $0.791_{\pm0.000}$ | $0.731_{\pm0.002}$ | $0.821_{\pm0.000}$ | $0.794_{\pm0.000}$ | $0.839_{\pm0.000}$ | $0.808_{\pm0.002}$ | $0.841_{\pm0.001}$ | $0.818_{\pm0.002}$ |
| ✓ | ✓ | $\mathbf{0.827}_{\pm0.093}$ | $\mathbf{0.847}_{\pm0.034}$ | $\mathbf{0.846}_{\pm0.084}$ | $\mathbf{0.902}_{\pm0.000}$ | $\mathbf{0.863}_{\pm0.000}$ | $\mathbf{0.952}_{\pm0.000}$ | $\mathbf{0.823}_{\pm0.000}$ | $\mathbf{0.858}_{\pm0.000}$ | $0.827_{\pm0.000}$ |

Table 3: Further ablation study of scCR on cell clustering measured by ARI. PRE, COM, and DEN denote the pre-imputation stage, the complete relation stage, and the denosing stage, respectively.

| PRE | COM | DEN | Baron Mouse | Zeisel | Baron Human |
|---|---|---|---|---|---|
| ✓ | ✗ | ✗ | $0.437 \pm 0.061$ | $0.682 \pm 0.024$ | $0.580 \pm 0.036$ |
| ✓ | ✓ | ✗ | $0.409 \pm 0.008$ | $0.732 \pm 0.001$ | $0.571 \pm 0.007$ |
| ✓ | ✗ | ✓ | $0.584 \pm 0.000$ | $0.822 \pm 0.000$ | $0.681 \pm 0.000$ |
| ✓ | ✓ | ✓ | $\mathbf{0.827 \pm 0.093}$ | $\mathbf{0.902 \pm 0.000}$ | $\mathbf{0.823 \pm 0.000}$ |

We now focus solely on the convergence of $\lim_{K\to\infty} \bar{\mathbf{x}}_u^{(a)}(K)$. When we unroll the recursion for the last $|\bar{\mathbf{x}}_u^{(a)}|$ rows,

$$
\begin{aligned}
\bar{\mathbf{x}}_u^{(a)}(K) &= \overline{\mathbf{A}}_{uk}^{(a)}\mathbf{x}_k^{(a)} + \overline{\mathbf{A}}_{uu}^{(a)}\bar{\mathbf{x}}_u^{(a)}(K-1) \\
&= \overline{\mathbf{A}}_{uk}^{(a)}\mathbf{x}_k^{(a)} + \overline{\mathbf{A}}_{uu}^{(a)}(\overline{\mathbf{A}}_{uk}^{(a)}\mathbf{x}_k^{(a)} + \overline{\mathbf{A}}_{uu}^{(a)}\bar{\mathbf{x}}_u^{(a)}(K-2)) \\
&= \ldots \\
&= \Big(\sum_{t=0}^{K-1}(\overline{\mathbf{A}}_{uu}^{(a)})^t\Big)\overline{\mathbf{A}}_{uk}^{(a)}\mathbf{x}_k^{(a)} + (\overline{\mathbf{A}}_{uu}^{(a)})^K\bar{\mathbf{x}}_u^{(a)}(0)
\end{aligned}
$$

By Lemma 1, $\lim_{K\to\infty}(\overline{\mathbf{A}}_{uu}^{(a)})^K = 0$. Therefore, $\lim_{K\to\infty}(\overline{\mathbf{A}}_{uu}^{(a)})^K\bar{\mathbf{x}}_u^{(a)}(0) = 0$, regardless of the initial state for $\bar{\mathbf{x}}_u^{(a)}(0)$. (we replace $\bar{\mathbf{x}}_u^{(a)}(0)$ with a zero column vector for simplicity.) Hence, our focus shifts to $\lim_{K\to\infty}(\sum_{t=0}^{K-1}(\overline{\mathbf{A}}_{uu}^{(a)})^t)\overline{\mathbf{A}}_{uk}^{(a)}\mathbf{x}_k^{(a)}$.

Given that Lemma 1 establishes $\rho(\overline{\mathbf{A}}_{uu}^{(a)}) < 1$, and Lemma 2 affirms the invertibility of $(\mathbf{I}_{uu} - \overline{\mathbf{A}}_{uu}^{(a)})^{-1}$, the geometric series converges as follows

$$
\lim_{K\to\infty} \bar{\mathbf{x}}_u^{(a)}(K) = \lim_{K\to\infty}\Big(\sum_{t=0}^{K-1}(\overline{\mathbf{A}}_{uu}^{(a)})^t\Big)\overline{\mathbf{A}}_{uk}^{(a)}\mathbf{x}_k^{(a)} = (\mathbf{I}_{uu} - \overline{\mathbf{A}}_{uu}^{(a)})^{-1}\overline{\mathbf{A}}_{uk}^{(a)}\mathbf{x}_k^{(a)}.
$$

In conclusion, the recursion in FP converges. □

## C  Ablation Study

We conducted an ablation study to analyze the effectiveness of each component in scCR. We performed cell clustering on the Baron Mouse, Zeisel, and Baron Human datasets. As shown in Table 2, both concatenation and standardization contributed to performance improvement, and the combination of the two components led to significant performance improvement.

We conducted an additional ablation study to assess the effectiveness of each stage of scCR. Table 3 presents the results of the ablation study in terms of cell clustering, measured by ARI. As shown in the table, adding the complete relation stage and the denoising stage significantly improved performance compared to using only the pre-imputation stage. These results confirm that the complete relation and denoising stages contributed substantially to the high performance of scCR, underscoring the well-founded design of our approach.

Table 4: Performance on dropout recovery under Missing Not At Random (MNAR) settings, measured by RMSE.

| Dataset | scFP | scBFP | scCR (ours) |
|---|---|---|---|
| Baron Mouse | $0.517 \pm 0.000$ | $0.465 \pm 0.000$ | $\mathbf{0.304 \pm 0.000}$ |
| Pancreas | $0.537 \pm 0.001$ | $0.506 \pm 0.001$ | $\mathbf{0.352 \pm 0.000}$ |
| Mouse Bladder | $0.374 \pm 0.000$ | $0.374 \pm 0.001$ | $\mathbf{0.170 \pm 0.000}$ |
| Zeisel | $0.580 \pm 0.001$ | $0.538 \pm 0.000$ | $\mathbf{0.489 \pm 0.000}$ |
| Worm Neuron | $0.330 \pm 0.000$ | $0.190 \pm 0.000$ | $\mathbf{0.049 \pm 0.000}$ |
| Baron Human | $0.493 \pm 0.000$ | $0.475 \pm 0.000$ | $\mathbf{0.328 \pm 0.000}$ |

# D   Missing Not at Random Settings

Existing studies [11, 12] simulate dropout by randomly sampling non-zero values in a cell-gene matrix from a uniform distribution and setting them to zero (*i.e.*, missing completely at random (MCAR)). However, in real scRNA-seq data, dropouts occur more frequently in genes with low expression levels rather than those with high variance [34]. This is because the probability of capturing RNA transcripts of low-expression-level genes during sequencing is lower. Based on this dropout pattern, we selected the 1,000 genes with the lowest expression levels and simulated dropout only in these genes. We randomly sampled non-zero values of these genes from a uniform distribution and replaced the sampled values with zero (*i.e.*, missing not at random (MNAR)).

Table 4 presents the performance comparison under the aforementioned MNAR settings in terms of data recovery, measured by RMSE. We compared our scCR to the two most competitive baselines, scFP [11] and scBFP [12]. The dropout rate was set to 20% of the total number of values in the cell-gene matrix. As shown in the table, scCR outperformed the compared methods by significant margins in the realistic dropout settings, demonstrating the robustness of scCR in realistic scenarios. Considering realistic dropout simulation can help pre-assess the generalizability of techniques in practical scRNA-seq applications.

# E   Memory Usage Analysis

Table 5: Comparison of input and memory complexity. $\mathbf{X} \in \mathbb{R}^{G \times C}$ denote a cell-gene matrix, where $C$ and $G$ represent the number of cells and genes, respectively. $\mathbf{A}^{cell} \in \mathbb{R}^{C \times C}$ and $\mathbf{A}^{gene} \in \mathbb{R}^{G \times G}$ denote cell-cell and gene-gene adjacency matrices, respectively. $\theta$ denotes trainable parameters. $B$ represents the batch size for batch-wise $k$-NN graph construction.

| Method | Input | Big-O |
|---|---|---|
| scTAG | $\mathbf{X}, \mathbf{A}^{cell}, \theta$ | $O(GC) + O(\mathcal{E}_{cell}) + O(\theta)$ |
| DCA | $\mathbf{X}, \theta$ | $O(GC) + O(\theta)$ |
| AutoClass | $\mathbf{X}, \theta$ | $O(GC) + O(\theta)$ |
| scGNN 2.0 | $\mathbf{X}, \theta$ | $O(GC) + O(\mathcal{E}_{cell}) + O(\theta)$ |
| scGCL | $\mathbf{X}, \mathbf{A}^{cell}, \theta$ | $O(GC) + O(\mathcal{E}_{cell}) + O(\theta)$ |
| MAGIC | $\mathbf{X}, \mathbf{A}^{cell}$ | $O(GC) + O(\mathcal{E}_{cell})$ |
| scFP | $\mathbf{X}, \mathbf{A}^{cell}$ | $O(BC) + O(\mathcal{E}_{cell})$ |
| scBFP | $\mathbf{X}, \mathbf{A}^{cell}, \mathbf{A}^{gene}$ | $O(BC) + O(\mathcal{E}_{gene}) + O(BG) + O(\mathcal{E}_{cell})$ |
| scCR (Ours) | $\mathbf{X}, \mathbf{A}^{cell}, \mathbf{A}^{gene}$ | $O(BC) + O(\mathcal{E}_{gene}) + O(BG) + O(\mathcal{E}_{cell})$ |

We analyzed the memory complexity of all methods used in this paper and conducted additional experiments to examine the memory usage of our scCR. Table 5 compares the input and memory complexity of scCR with other state-of-the-art methods. To alleviate the high memory demands during $k$-NN graph construction, we adopted the batch-wise $k$-NN graph construction strategy from [12]. When constructing $k$-NN graphs among genes, we divided the genes into batches of size $B$ and computed $k$-nearest neighbors for each batch. We applied the same batch-wise strategy when constructing $k$-NN graphs among cells. This approach reduces memory requirements by avoiding the need to store distances between all points in the entire dataset simultaneously. Specifically, in the memory complexity of scCR, batch-wise $k$-NN graph construction changes $O(G^2)$ and $O(C^2)$ to $O(BG)$ and $O(BC)$, respectively. Consequently, batch-wise $k$-NN graph construction enables

Table 6: Memory usage of scCR for different datasets, measured by gigabytes (GB).

| | Baron Mouse | Pancreas | Mouse Bladder | Zeisel | Worm Neuron | Baron Human |
|---|---|---|---|---|---|---|
| Memory usage (GB) | 1.811 | 1.837 | 1.957 | 2.037 | 1.927 | 3.861 |

processing of large datasets that would otherwise be infeasible due to memory constraints. Moreover, scCR does not require any trainable parameters, unlike other deep-learning-based methods.

We further measured the memory usage of scCR across various datasets, as shown in Table 6. The results in the table indicate that the advantages of scCR extend beyond its superior performance and time efficiency, showcasing its scalability as well.

# F  Hyperparameter Sensitivity

Table 7: Performance of scCR on cell clustering measured by ARI for different values of $\alpha$.

| $\alpha$ | Baron Mouse | Zeisel | Baron Human |
|---|---|---|---|
| 0.01 | $0.627 \pm 0.000$ | $0.903 \pm 0.000$ | $0.823 \pm 0.000$ |
| 0.05 (used) | $0.827 \pm 0.093$ | $0.902 \pm 0.000$ | $0.823 \pm 0.000$ |
| 0.1 | $0.727 \pm 0.141$ | $0.904 \pm 0.000$ | $0.824 \pm 0.000$ |
| 0.5 | $0.701 \pm 0.000$ | $0.901 \pm 0.000$ | $0.681 \pm 0.000$ |
| 0.9 | $0.448 \pm 0.001$ | $0.825 \pm 0.000$ | $0.683 \pm 0.001$ |

Table 8: Performance of scCR on cell clustering measured by ARI for different values of $\beta$.

| $\beta$ | Baron Mouse | Zeisel | Baron Human |
|---|---|---|---|
| 0.1 | $0.440 \pm 0.046$ | $0.659 \pm 0.058$ | $0.553 \pm 0.040$ |
| 0.5 | $0.476 \pm 0.049$ | $0.724 \pm 0.000$ | $0.560 \pm 0.001$ |
| 0.9 | $0.509 \pm 0.004$ | $0.740 \pm 0.000$ | $0.657 \pm 0.000$ |
| 0.95 | $0.498 \pm 0.002$ | $0.910 \pm 0.000$ | $0.666 \pm 0.011$ |
| 0.99 (used) | $0.827 \pm 0.093$ | $0.902 \pm 0.000$ | $0.823 \pm 0.000$ |
| 0.999 | $0.925 \pm 0.000$ | $0.900 \pm 0.000$ | $0.819 \pm 0.000$ |

Table 9: Performance of scCR on cell clustering measured by ARI for different values of $\gamma$.

| $\gamma$ | Baron Mouse | Zeisel | Baron Human |
|---|---|---|---|
| 0.001 | $0.927 \pm 0.001$ | $0.902 \pm 0.000$ | $0.824 \pm 0.000$ |
| 0.01 (used) | $0.827 \pm 0.093$ | $0.902 \pm 0.000$ | $0.823 \pm 0.000$ |
| 0.05 | $0.635 \pm 0.000$ | $0.903 \pm 0.000$ | $0.822 \pm 0.000$ |
| 0.1 | $0.595 \pm 0.000$ | $0.903 \pm 0.000$ | $0.667 \pm 0.001$ |
| 0.5 | $0.469 \pm 0.003$ | $0.740 \pm 0.000$ | $0.627 \pm 0.011$ |
| 0.9 | $0.409 \pm 0.009$ | $0.749 \pm 0.000$ | $0.586 \pm 0.006$ |

Table 10: Performance of scCR on cell clustering measured by ARI for different values of $k$.

| $k$ | Baron Mouse | Zeisel | Baron Human |
|---|---|---|---|
| 1 | $0.628 \pm 0.000$ | $0.905 \pm 0.000$ | $0.827 \pm 0.000$ |
| 2 (used) | $0.827 \pm 0.093$ | $0.902 \pm 0.000$ | $0.823 \pm 0.000$ |
| 3 | $0.631 \pm 0.000$ | $0.903 \pm 0.000$ | $0.819 \pm 0.000$ |
| 5 | $0.625 \pm 0.002$ | $0.902 \pm 0.001$ | $0.818 \pm 0.000$ |
| 10 | $0.621 \pm 0.000$ | $0.903 \pm 0.000$ | $0.818 \pm 0.000$ |
| 15 | $0.630 \pm 0.000$ | $0.902 \pm 0.000$ | $0.817 \pm 0.000$ |

We conducted additional experiments to provide a comprehensive analysis of the impact of different hyperparameters, including $\alpha$, $\beta$, $\gamma$, and $k$, on the performance of scCR. We report ARI in cell clustering on three datasets by varying $\alpha$, $\beta$, $\gamma$, and $k$ within the ranges of {0.01, 0.05, 0.1, 0.5,

0.9}, {0.1, 0.5, 0.9, 0.95, 0.99, 0.999}, {0.001, 0.01, 0.05, 0.1, 0.5, 0.9}, and {1, 2, 3, 5, 10, 15}, respectively. When varying a target parameter, other hyperparameters were fixed to their default settings. Table 7, Table 8, Table 9, and Table 10 illustrate how these hyperparameter choices impact scCR performance. As shown in the tables, the values of $\alpha$, $\beta$, $\gamma$, and $k$ used in this study generally resulted in strong performance.

In terms of sensitivity, when the runner-up ARI scores are $0.660\pm0.00$, $0.848\pm0.00$, and $0.677\pm0.00$ for Baron Mouse, Zeisel, Baron Human, respectively, scCR demonstrated robustness against hyperparameter variations. Specifically, $\alpha \in \{0.05, 0.1, 0.5\}$, $\beta \in \{0.99, 0.999\}$, and $\gamma \in \{0.001, 0.01\}$ yielded state-of-the-art performance across the datasets. For $k$, scCR showed strong performance across all values, except in the case of Baron Human. Additionally, some parameter adjustments led to performance surpassing that of the default settings. However, given the unsupervised nature of single-cell analysis, we retain default hyperparameter settings that generally perform well.

# G  Experimental Details

## G.1  Implementation Details

We conducted all the experiments on a single NVIDIA GeForce RTX 2080 Ti GPU and an Intel Core I5-6600 CPU at 3.30 Hz. The number of neighbors ($k$) in cell-cell and gene-gene $k$-NN graphs were set to 15 and 2, respectively. The total number of propagation steps $K$ was set to 40 for both cell-cell and gene-gene FP. We set $\alpha$, $\beta$, and $\gamma$ to 0.05, 0.99, and 0.01, respectively. We found that the choice between row-stochastic normalization and symmetric normalization applied to $\overline{\mathbf{A}}^{cell(3)}$ within Soft FP [11] affected performance, and we reported the best result. For dropout recovery, we excluded the denoising stage (*i.e.*, $\gamma = 1$).

## G.2  Datasets

For our experiments, we utilized six real scRNA-seq datasets, including Baron Mouse [25], Pancreas [26], Mouse Bladder [27], Zeisel [2], Worm Neuron [28], and Baron Human [25]. Table 11 summarizes the dataset statistics. We downloaded all the datasets from the GitHub repository[2] for [11]. These publicly available datasets and the repository have no public declaration of license.

We leveraged a commonly used pre-processing procedure for scRNA-seq data as described in recent studies [11, 12]. We performed minimal quality control (QC) using SCANPY [35], a toolkit for scRNA-seq analyses. Cells and genes exhibiting no gene expression (*i.e.*, with all zero values) were removed from a given cell-gene matrix. We retained the 2,000 genes with the highest variance in each dataset. We then normalized each cell by total counts over all genes to ensure that every cell had an equal total count of 1.0. That is, every row vector was divided by its library size, which is the sum of its values. After scaling by the median library size, $\log(x + 1)$ transformation was applied to all values in the cell-gene matrix, resulting in a pre-processed cell-gene matrix.

Table 11: Dataset statistics.

| Dataset | Protocol | #Cells | #Genes | #Cell Type |
|---|---|---|---|---|
| Baron Mouse | inDrop | 1,886 | 14,861 | 13 |
| Pancreas | inDrop | 1,937 | 15,575 | 14 |
| Mouse Bladder | Microwell-seq | 2,746 | 19,771 | 16 |
| Zeisel | STRT-seq UMI | 3,005 | 19,972 | 7 |
| Worm Neuron | sci-RNA-seq | 4,186 | 13,488 | 10 |
| Baron Human | inDrop | 8,569 | 17,499 | 14 |

### G.3  Baselines

For all the baselines, we used the code released by the author of the respective papers. Table 12 shows the URL links for the baselines. scTAG and scGNN 2.0 are under the MIT license. The licenses of DCA, AutoClass, and MAGIC are Apache-2.0, GPL-3.0, and GPL-2.0, respectively. The code for scGCL, scFP, and scBFP has no public declaration of license. For each baseline, we adhered the hyperpameter/parameter setting in the released code or its respective paper.

Table 12: URL links for baselines.

| Baseline | URL Link |
|---|---|
| scTAG | https://github.com/Philyzh8/scTAG |
| DCA | https://github.com/theislab/dca |
| AutoClass | https://github.com/datapplab/AutoClass |
| scGNN 2.0 | https://github.com/OSU-BMBL/scGNN2.0 |
| scGCL | https://github.com/zehaoxiong123/scGCL |
| MAGIC | https://github.com/KrishnaswamyLab/MAGIC |
| scFP | https://github.com/Junseok0207/scFP |
| scBFP | https://github.com/Junseok0207/scBFP |

### G.4  Evaluation Metrics

#### G.4.1  Clustering

Higher ARI, NMI, and CA indicate better performance in cell clustering.

**ARI.** The Adjusted Rand Index (ARI) is the corrected-for-chance version of the Rand Index (RI). RI is computed as follows:

$$RI = \frac{TP + TN}{\binom{N}{2}} \tag{12}$$

where $TP$ is the number of true positives and $TN$ is the number of true negatives. True positive indicates the number of cell pairs correctly assigned to the same cluster, and $TP$ indicates the number of cell pairs correctly assigned to different clusters. ARI can be calculated as follows:

$$ARI = \frac{RI - \mathbb{E}(RI)}{\max(RI) - \mathbb{E}[RI]} \tag{13}$$

While RI produces a value between 0 and 1, ARI can produce negative values if the index is less than the expected index.

**NMI.** Normalized Mutual Information (NMI) is a normalization of the Mutual Information (MI) score to scale the scores between 0 and 1. NMI is computed as follows:

$$NMI = \frac{2 \times I(S;C)}{H(S) + H(C)} \tag{14}$$

where $S$ is ground-truth cell types, $I(\cdot, \cdot)$ is the mutual information between two input distributions, and $H(\cdot)$ is the entropy function. Here, all logs are base-2. Higher NMI indicates the distribution of predicted cluster distribution is more similar to ground-truth cell type distribution.

**CA.** Clustering Accuracy is calculated as follows:

$$CA = \max_{m} \frac{\sum_{i=1}^{N} \mathbb{1}_{s_i = m(c_i)}}{N} \tag{15}$$

where $s_i$ is the ground-truth cell type of the $i$-th cell, $c_i$ is predicted cluster assignment of the $i$-th cell, and $m(\cdot)$ is the matching function responsible for mapping predicted cluster assignments to ground-truth cell types.

#### G.4.2  Recovery

Lower Median L1 Distance and RMSE indicate better performance in data recovery. Consider two set $\mathcal{X} = \{x_1, \ldots, x_n\}$ and $\mathcal{Y} = \{y_1, \ldots, y_n\}$, where $\mathcal{X}$ is the set of imputed values and $\mathcal{Y}$ is the set of their ground-truth values.

**Median L1 Distance.** Median L1 Distance is calculated as follows:

$$Median\ L1\ Distance = median(|x_1 - y_1|, \ldots, |x_n - y_n|). \qquad (16)$$

**RMSE.** Root Mean Square Error (RMSE) is computed as follows:

$$RMSE(\mathcal{X}, \mathcal{Y}) = \sqrt{\frac{\sum_{i=1}^{N}(x_i - y_i)^2}{N}} \qquad (17)$$

# H    Additional Experimental Results

## H.1    Varying Scales across Genes

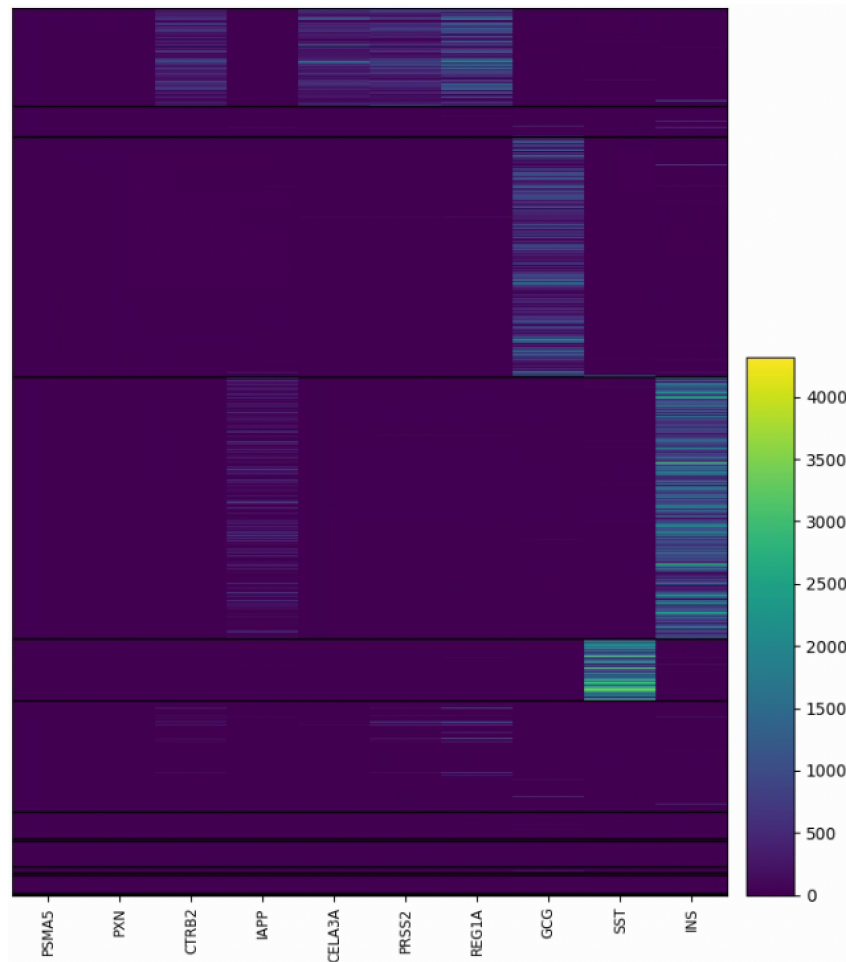

Figure 9: An heatmap of the cell-gene matrix in the Baron Human dataset. We randomly selected 10 genes (columns).

## H.2 Dropout Recovery

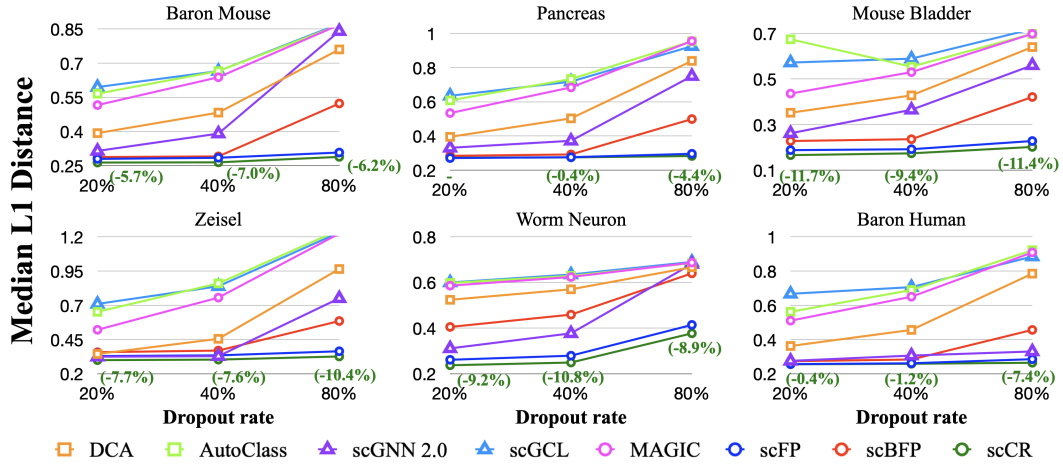

Figure 10: Performance on dropout recovery, measured by L1 Median Distance. Figures highlighted in green indicate reduction rates from the most competitive baseline at each setting.

